# Space-Variant Single-Image Blind Deconvolution for Removing Camera Shake

**Stefan Harmeling, Michael Hirsch, and Bernhard Schölkopf**
Max Planck Institute for Biological Cybernetics, Tübingen, Germany
`firstname.lastname@tuebingen.mpg.de`

## Abstract

Modelling camera shake as a space-invariant convolution simplifies the problem of removing camera shake, but often insufficiently models actual motion blur such as those due to camera rotation and movements outside the sensor plane or when objects in the scene have different distances to the camera. In an effort to address these limitations, (i) we introduce a taxonomy of camera shakes, (ii) we build on a recently introduced framework for space-variant filtering by Hirsch et al. and a fast algorithm for single image blind deconvolution for space-invariant filters by Cho and Lee to construct a method for blind deconvolution in the case of space-variant blur, and (iii), we present an experimental setup for evaluation that allows us to take images with real camera shake while at the same time recording the space-variant point spread function corresponding to that blur. Finally, we demonstrate that our method is able to deblur images degraded by spatially-varying blur originating from real camera shake, even without using additionally motion sensor information.

## 1 Introduction

Camera shake is a common problem of handheld, longer exposed photographs occurring especially in low light situations, e.g., inside buildings. With a few exceptions such as panning photography, camera shake is unwanted, since it often destroys details and blurs the image. The effect of a particular camera shake can be described by a linear transformation on the sharp image, i.e., the image that would have been recorded using a tripod. Denoting for simplicity images as column vectors, the recorded blurry image $y$ can be written as a linear transformation of the sharp image $x$, i.e., as $y = Ax$, where $A$ is an unknown matrix describing the camera shake. The task of blind image deblurring is to recover $x$ given only the blurred image $y$, but not $A$.

**Main contributions.** (i) We present a taxonomy of camera shakes; (ii) we propose an algorithm for deblurring space-variant camera shakes; and (iii) we introduce an experimental setup that allows to simultaneously record images blurred by real camera shake and an image of the corresponding spatially varying point spread functions (PSFs).

**Related work.** Our work combines ideas of three papers: (i) Hirsch et al's work [1] on *efficient space-variant* filtering, (ii) Cho and Lee's work [2] on *single frame* blind deconvolution, and (iii) Krishnan and Fergus's work [3] on *fast* non-blind deconvolution.

Previous approaches to single image blind deconvolution have dealt only with space-invariant blurs. This includes the works of Fergus et al. [4], Shan et al. [5], as well as Cho and Lee [2] (see Kundur and Hatzinakos [6] and Levin et al. [7] for overviews and further references).

Tai et al. [8] represent space-variant blurs as projective motion paths and propose a non-blind deconvolution method. Shan et al. [9] consider blindly deconvolving rotational object motion, yielding a particular form of space-variant PSFs. Blind deconvolution of space-variant blurs in the context

of star fields has been considered by Bardsley et al. [10]. Their method estimates PSFs separately (and not simultaneously) on image patches using phase diversity, and deconvolves the overall image using [11]. Joshi et al. [12] recently proposed a method that estimates the motion path using inertial sensors, leading to high-quality image reconstructions.

There exists also some work for images in which different segments have different blur: Levin [13] and Cho et al. [14] segment images into layers where each layer has a different motion blur. Both approaches consider uniform object motion, but not non-uniform ego-motion (of the camera). Hirsch et al. [1] require multiple images to perform blind deconvolution with space-variant blur, as do Šorel and Šroubek [15].

## 2   A taxonomy of camera shakes

Camera shake can be described from two perspectives: (i) how the PSF varies across the image, i.e., how point sources would be recorded at different locations on the sensor, and (ii) by the trajectory of the camera and how the depth of the scene varies. Throughout this discussion we assume the scene to be static, i.e., only the camera moves (only ego-motion), and none of the photographed objects (no object motion).

**PSF variation across the image.** We distinguish three classes:

- **Constant:** The PSF is constant across the image. In this case the linear transformation is a convolution matrix. Most algorithms for blind deconvolution are restricted to this case.
- **Smooth:** The PSF is smoothly varying across the image. Here, the linear transformation is no longer a convolution matrix, but a more general framework is needed such as the smoothly space-varying filters in the multi-frame method of Hirsch et al. [1]. For this case, our paper proposes an algorithm for *single* image deblurring.
- **Segmented:** The PSF varies smoothly within segments of the image, but between segments it may change abruptly.

**Depth variation across the scene.** The depth in a scene, i.e., the distance of the camera to objects at different locations in the scene, can be classified into three categories:

- **Constant:** All objects have the same distance to the camera. Example: photographing a picture hanging on the wall.
- **Smooth:** The distance to the camera is smoothly varying across the scene. Example: photographing a wall at an angle.
- **Segmented:** The scene can be segmented into different objects each having a different distance to the camera. Example: photographing a scene with different objects partially occluding each other.

**Camera trajectories.** The motion of the camera can be represented by a six dimensional trajectory with three spatial and three angular coordinates. We denote the two coordinates inside the sensor plane as $a$ and $b$, the coordinate corresponding to the distance to the scene as $c$. Furthermore, $\alpha$ and $\beta$ describe the camera tilting up/down and left/right, and $\gamma$ the camera rotation around the optical axis.

It is instructive to picture how different trajectories correspond to different PSF variations in different depth situations. Exemplarily we consider the following trajectories:

- **Pure shift:** The camera moves inside the sensor plane without rotation; only $a$ and $b$ vary.
- **Rotated shift:** The camera moves inside the sensor plane with rotation; $a$, $b$, and $\gamma$ vary.
- **Back and forth:** The distance between camera and scene is changing; only $c$ varies.
- **Pure tilt:** The camera is tilted up and down and left and right; only $\alpha$ and $\beta$ vary.
- **General trajectory:** All coordinates might vary as a function of time.

Table 1 shows all possible combinations. Note that only "pure shifts" in combination with "constant depths" lead to a constant PSF across the image, which is the case most methods for camera unshaking are proposed for. Thus, extending blind deconvolution to smoothly space-varying PSFs can increases the range of possible applications. Furthermore, we see that for segmented scenes, camera shake usually leads to blurs that are non-smoothly changing across the image. Even though

| | Pure shift | Rotated shift | Back and forth | Pure tilt | General trajectory |
|---|---|---|---|---|---|
| Constant depth | constant | smooth | smooth | smooth | smooth |
| Smooth depth | smooth | smooth | smooth | smooth | smooth |
| Segmented depth | segmented | segmented | segmented | segmented | segmented |

Table 1: How the PSF varies for different camera trajectories and for different depth situations.

in this case the model of smoothly varying PSFs is incorrect, it might still lead to better results than constant PSFs.

## 3 Smoothly varying PSF as Efficient Filter Flow

To obtain a generalized image deblurring method we represent the linear transformation $y = Ax$ by the recently proposed efficient filter flow (EFF) method of Hirsch et al. [1] that can handle smoothly varying PSFs. For convenience, we briefly describe EFF, using the notation and results from [1].

**Space-invariant filters.** As our starting point we consider *space-invariant* filters (aka convolutions), which are an efficient, but restrictive class of linear transformations. We denote by $y$ the recorded image, represented as a column vector of length $m$, and by $a$ a column vector of length $k$, representing the space-invariant PSF, and by $x$ the true image, represented as a column vector of length $n = m + k - 1$ (we consider the *valid* part of the convolution). Then the usual convolution can be written as $y_i = \sum_{j=0}^{k-1} a_j x_{i-j}$ for $0 \leq i < m$. This transformation is linear in $x$, and thus an instance of the general linear transformation $y = Ax$, where the column vector $a$ parametrizes the transformation matrix $A$. Furthermore, the transformation is linear in $a$, which implies that there exists a matrix $X$ such that $y = Ax = Xa$. Using fast Fourier transforms (FFTs), these matrix-vector-multiplications (MVMs) can be calculated in $O(n \log n)$.

**Space-variant filters.** Although being efficient, the (space-invariant) convolution applies only to camera shakes which are pure shifts of flat scenes. This is generalized to *space-variant* filtering by employing Stockham's overlap-add (OLA) trick [16]. The idea is (i) to cover the image with *overlapping* patches, (ii) to apply to each patch a different PSF, and (iii) to add the patches to obtain a single large image. The transformation can be written as

$$y_i = \sum_{r=0}^{p-1} \sum_{j=0}^{k-1} a_j^{(r)} w_{i-j}^{(r)} x_{i-j} \text{ for } 0 \leq i < m \text{ where } \sum_{r=0}^{p-1} w_i^{(r)} = 1 \text{ for } 0 \leq i < m. \qquad (1)$$

Here, $w^{(r)} \geq 0$ smoothly fades the $r$-th patch in and masks out the others. Note that at each pixel the sum of the weights must sum to one.

Note that this method does *not* simply apply a different PSF to different image regions, but instead yields a different PSF for *each* pixel. The reason is that usually, the patches are chosen to overlap at least 50%, so that the PSF at a pixel is a certain linear combination of several filters, where the weights are chosen to smoothly blend filters in and out, and thus the PSF tends to be different at each pixel. Fig. 1 shows that a PSF array as small as $3 \times 3$, corresponding to $p = 9$ and nine overlapping patches (right panel of the bottom row), can parametrize smoothly varying blurs (middle column) that closely mimic real camera shake (left column).

**Efficient implementation.** As is apparent from Eq. (1), EFF is linear in $x$ and in $a$, the vector obtained by stacking $a^{(0)}, \ldots, a^{(p-1)}$. This implies that there exist matrices $A$ and $X$ such that $y = Ax = Xa$. Using Stockham's ideas [16] to speed-up large convolutions, Hirsch et al. derive expressions for these matrices, namely

$$A = Z_y^{\mathsf{T}} \sum_{r=0}^{p-1} C_r^{\mathsf{T}} F^{\mathsf{H}} \operatorname{Diag}(F Z_a a^{(r)}) F C_r \operatorname{Diag}(w^{(r)}), \qquad (2)$$

$$X = Z_y^{\mathsf{T}} \sum_{r=0}^{p-1} C_r^{\mathsf{T}} F^{\mathsf{H}} \operatorname{Diag}(F C_r \operatorname{Diag}(w^{(r)}) x) F Z_a B_r, \qquad (3)$$

where $\operatorname{Diag}(w^{(r)})$ is the diagonal matrix with vector $w^{(r)}$ along its diagonal, $C_r$ is a matrix that crops out the $r$-th patch, $F$ is the discrete Fourier transform matrix, $Z_a$ is a matrix that zero-pads

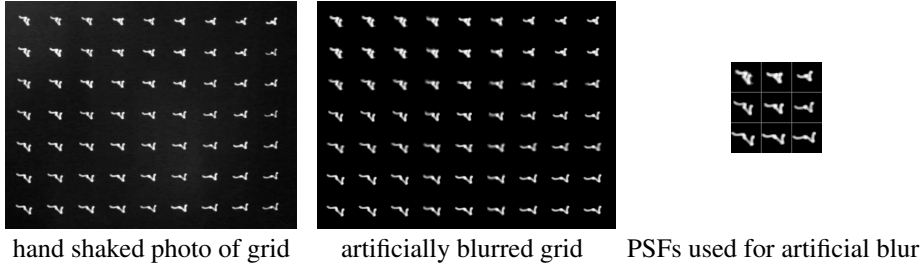

hand shaked photo of grid     artificially blurred grid     PSFs used for artificial blur

Figure 1: A small set of PSFs can parametrize smoothly varying blur: (left) grid photographed with real camera shake, (middle) grid blurred by the EFF framework parametrized by nine PSFs (right).

$a^{(r)}$ to the size of the patch, $F^{\mathsf{H}}$ performs the inverse Fourier transform, $Z_y^{\mathsf{T}}$ chops out the valid part of the space-variant convolution.

Reading Eqs. (2) and (3) forward and backward yields efficient implementations for $A$, $A^{\mathsf{T}}$, $X$, and $X^{\mathsf{T}}$ with running times $O(n \log q)$ where $q$ is the patch size, see [1] for details. The overlap increases the computational cost by a constant factor and is thus omitted. The EFF framework thus implements space-variant convolutions which are as efficient to compute as space-invariant convolutions, while being much more expressive.

Note that each of the MVMs with $A$, $A^{\mathsf{T}}$, $X$, and $X^T$ is needed for blind deconvolution: $A$ and $A^{\mathsf{T}}$ for the estimation of $x$ given $a$, and $X$ and $X^T$ for the estimation of $a$.

## 4 Blind deconvolution with smoothly varying PSF

We now outline a single image blind deconvolution algorithm for space-variant blur, generalizing the method of Cho and Lee [2], that aims to recover a sharp image in two steps: (i) first estimate the parameter vector $a$ of the EFF transformation, and (ii) then perform space-variant non-blind deconvolution by running a generalization of Krishnan and Fergus' algorithm [3].

**(i) Estimation of the linear transformation:** initializing $x$ with the blurry image $y$, the estimation of the linear transformation $A$ parametrized as an EFF, is performed by iterating over the following four steps:

- **Prediction step:** remove noise in flat regions of $x$ by edge-preserving bilateral filtering and overemphasize edges by shock filtering. To counter enhanced noise by shock filtering, we apply spatially adaptive gradient magnitude thresholding.
- **PSF estimation step:** update the PSFs given the blurry image $y$ and the current estimate of the predicted $x$, using only the gradient images of $x$ (resulting in a preconditioning effect) and enforcing smoothness between neighboring PSFs.
- **Propagation step:** identify regions of poorly estimated PSFs and replace them with neighboring PSFs.
- **Image estimation step:** update the current deblurred image $x$ by minimizing a least-squares cost function using a smoothness prior on the gradient image.

**(ii) Non-blind deblurring:** given the linear transformation we estimate the final deblurred image $x$ by alternating between the following two steps:

- **Latent variable estimation:** estimate latent variables regularized with a sparsity prior that approximate the gradient of $x$. This can be efficiently solved with look-up tables, see "w sub-problem" of [3] for details.
- **Image estimation step:** update the current deblurred image $x$ by minimizing a least-squares cost function while penalizing the Euclidean norm of the gradient image to the latent variables of the previous step, see "x sub-problem" of [3] for details.

The steps of (i) are repeated seven times on each scale of a multi-scale image pyramid. We always start with flat PSFs of size $3 \times 3$ pixels and the correspondingly downsampled observed image. For up- and downsampling we employ a simple linear interpolation scheme. The resulting PSFs in $a$

and the resulting image $x$ at each scale are upsampled and initialize the next scale. The final output of this iterative procedure are the PSFs that parametrize the spatially varying linear transformation.

Having obtained an estimate for the linear transformation in form of an array of PSFs, the alternating steps of (ii) perform space variant non-blind deconvolution of the recorded image $y$ using a natural image statistics prior (as in [13]). To this end, we adapt the recently proposed method of Krishnan and Fergus [3] to deal with linear transformations represented as EFF.

While our procedure is based on Cho and Lee's [2] and Krishnan and Fergus' [3] methods for space-invariant single blind deconvolution, it differs in several important aspects which we presently explain.

**Details of the Prediction step.** The prediction step of Cho and Lee [2] is a clever trick to avoid the nonlinear optimizations which would be necessary if the image features emphasized by the nonlinear filtering operations (namely shock and bilateral filtering and gradient magnitude thresholding) would have to be implemented by an image prior on $x$. Our procedure also profits from this trick and we set the hyper-parameters exactly as Cho and Lee do (see [2] for details on the nonlinear filtering operations). However, we note that for linear transformations represented as EFF, the gradient thresholding must be applied spatially adaptive, i.e., on each patch separately. This is necessary because otherwise a large gradient in some region might totally wipe out the gradients in regions that are less textured, leading to poor PSF estimates in those regions.

**Details on the PSF estimation step.** Given the thresholded gradient images of the nonlinear filtered image $x$ as the output of the prediction step, the PSF estimation minimizes a regularized least-squares cost function,

$$\sum_z \|\partial_z y - A \partial_z x\|^2 + \lambda \|a\|^2 + \nu g(a), \tag{4}$$

where $z$ ranges over the set $\{h, v, hh, vv, hv\}$, i.e., the first and second, horizontal and vertical derivatives of $y$ and $x$ are considered. Omitting the zeroth derivative (i.e., the images $x$ and $y$ themselves) has a preconditioning effect as discussed in Cho and Lee [2]. Matrix $A$ depends on the vector of PSFs $a$ as well. For the EFF framework we added the regularization term $g(a)$ which encourages similarity between neighboring PSFs,

$$g(a) = \sum_{r=0}^{p-1} \sum_{s \in \mathcal{N}(r)} \|a^{(r)} - a^{(s)}\|^2, \tag{5}$$

where $s \in \mathcal{N}(r)$ if patches $r$ and $s$ are neighbors.

**Details on the Propagation step.** Since high-frequency information, i.e. image details are required for PSF estimation, for images with less structured areas (such as sky) we can not estimate reasonable PSFs everywhere. The problem stems from the finding that even though some area might be less informative about the local PSF, it can look blurred, and thus would require deconvolution. These areas are identified by thresholding the entropy of the corresponding PSFs (similar to Šorel and Šroubek [15]). The rejected PSFs are replaced by the average of their neighboring PSFs. Since there might be areas for which the neighboring PSFs have been rejected as well, we perform a simple recursive procedure which propagates the accepted PSFs to the rejected ones.

**Details on the Image estimation step.** In both Cho and Lee's and also in Krishnan and Fergus' work, the image estimation step involves *direct deconvolution* which corresponds to a simple pixelwise divison of the blurry image by the zero-padded PSF in Fourier domain. Unfortunately, a direct deconvolution does not exist in general for linear transformations represented as EFF, since it involves summations over patches. However, we can replace the direct deconvolution by an optimization of some regularized least-squares cost function $\|y - Ax\|^2 + \alpha \|\nabla x\|^p$.

While estimating the linear transformation in (i), the regularizer is Tikhonov on the gradient image, i.e., $p = 2$. As the estimated $x$ is subsequently processed in the prediction step, one might consider regularization redundant in the image estimation step of (i). However, the regularization is crucial for suppressing ringing due to insufficient estimation of $a$. In (ii) during the final non-blind deblurring procedure we employ a sparsity prior for $x$ by choosing $p = 1/2$.

The main difference in the image estimation steps to [2] and [3] is that the linear transformation $A$ is no longer a convolution but instead a space-variant filter implemented by the EFF framework.

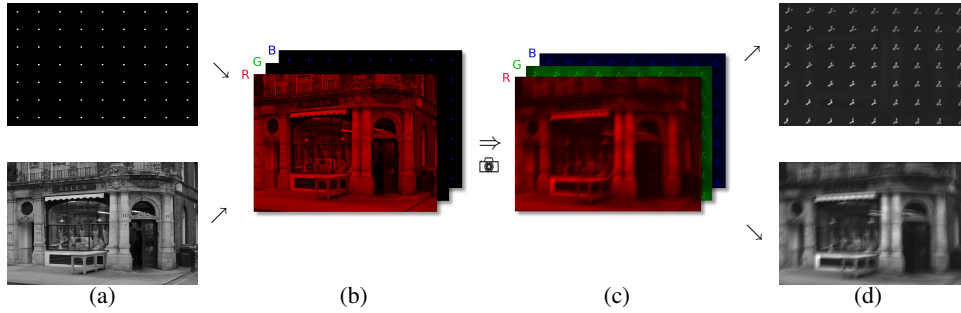

<div align="center">(a)            (b)            (c)            (d)</div>

Figure 2: How to simultaneously capture an image blurred with real camera shake and its space-varying PSF; (a) the true image and a grid of dots is combined to (b) an RBG image, that is (c) photographed with camera shake, and (d) split into blue and red channel to separate the PSF depicting the blur and the blurred image.

## 5   Experiments

We present results on several example images with space-variant blur, for which we are able to recover a deblurred image, while a state-of-the-art method for single image blind deconvolution does not. We begin by describing the image capture procedure.

**Capturing a gray scale image blurred with real camera shake along with the set of spatially varying PSFs.** The idea is to create a color image where the gray scale image is shown in the red channel, a grid of dots (for recording the PSFs) is shown in the blue channel, and the green channel is set to zero. We display the resulting RBG image on a computer screen and take a photo with real hand shake. We split the recorded raw image into the red and blue part. The red part only shows the image blurred with camera shake and the blue part shows the spatially varying PSFs that depict the effect of the camera shake. To avoid a Moiré effect the distance between the camera and the computer screen must be chosen carefully such that the discrete structure of the computer screen can not be resolved by the (discrete) image sensor of the camera. We verified that the spectral characteristics of the screen and the camera's Bayer array filters are such that there is no cross-talk, i.e., the blue PSFs are not visible in the red image. Fig. 2 shows the whole process.

**Three example images with real camera shake.** We applied our method, Cho and Lee's [2] method, and a custom patch-wise variant of Cho and Lee to three examples captured as explained above. For all experiments, photos were taken with a *hand-held* Canon EOS 1000D digital single lens reflex camera with a zoom lens (Canon zoom lens EF 24-70 mm 1:2.8 L USM). The exposure time was 1/4 second, the distance to the screen was about two meters. The input to the deblurring algorithm was only the red channel of the RAW file which we treat as if it were a captured gray-scale image. The image sizes are: vintage car $455 \times 635$, butcher shop $615 \times 415$, elephant $625 \times 455$.

To assess the accuracy of estimating the linear transformation (i.e., of step (i) in Sec. 4), we compare our estimated PSFs evaluated on a regular grid of dots to the true PSFs recorded in the blue channel during the camera shake. This comparison has been made for the vintage car example and is included in the supplementary material.

We compare with Cho and Lee's [2] method which we consider currently the state-of-art method for single image blind deconvolution. This method assumes space-invariant blurs, and thus we also compare to a modified version of this algorithm that is applied to the patches of our method and that finally blends the individually deblurred patches carefully to one final output image.

Fig 3 shows from top to bottom, the blurry captured image, the result of our method, Cho and Lee's [2] result, and a patch-wise variant of Cho and Lee. In our method we used for the linear transformation estimation step (step (i) in Sec. 4) for all examples the hyper-parameters detailed in [2]. Our additional hyper-parameters were set as follows: the regularization constant $\nu$ weighting the regularization term in cost function (4) that measures the similarity between neighboring PSFs is set to $5e4$ for all three examples. The entropy threshold for identifying poorly estimated PSFs is set to $0.7$, with the entropy normalized to range between zero and one. In all experiments, the size of a single PSF kernel is allowed to be $15 \times 15$ pixels. The space-variant blur was modelled for the

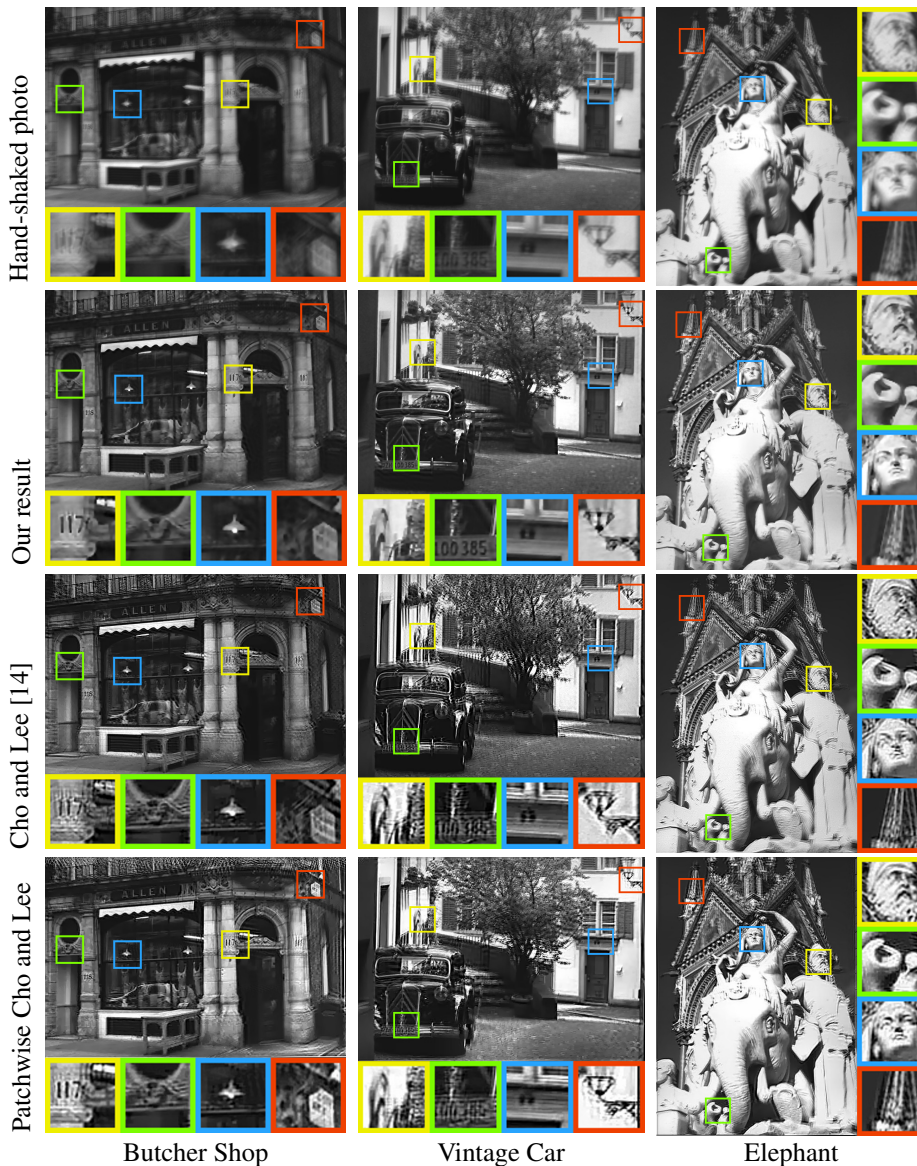

Figure 3: Deblurring results and comparison.

vintage car example by an array of $6 \times 7$ PSF kernels, for butcher shop by an array of $4 \times 6$ PSF kernels, and for the elephant by an array of $5 \times 6$ PSF kernels. These setting were also used for the patch-wise Cho and Lee variant. For the blending function $w^{(r)}$ in Eq. (1) we used a Bartlett-Hanning window with $75\%$ overlap in the vintage car example and $50\%$ in the butcher shop and elephant example. We choose for the vintage car a larger overlap to keep the patch size reasonably large. For the final non-blind deconvolution (step (ii) in Sec. 4) hyper-parameter $\alpha$ was set to $2e3$ and $p$ was set to $0.5$. On the three example images our algorithm took about 30 minutes for space-variant image restoration.

In summary, our experiments show that our method is able to deblur space-variant blurs that are too difficult for Cho and Lee's method. Especially, our results reveal greater detail and less restoration artifacts, especially noticeable in the regions of the closeup views. Interesting is the comparison with the patch-wise version of Cho and Lee: looking at the details (such as the house number 117 at the butcher shop, the licence plate of the vintage car, or the trunk of the elephant) our method is better. At the door frame in the vintage car image, we see that the patch-wise version of Cho and Lee has alignment problems. Our experience was that this gets more severe for larger blur kernels.

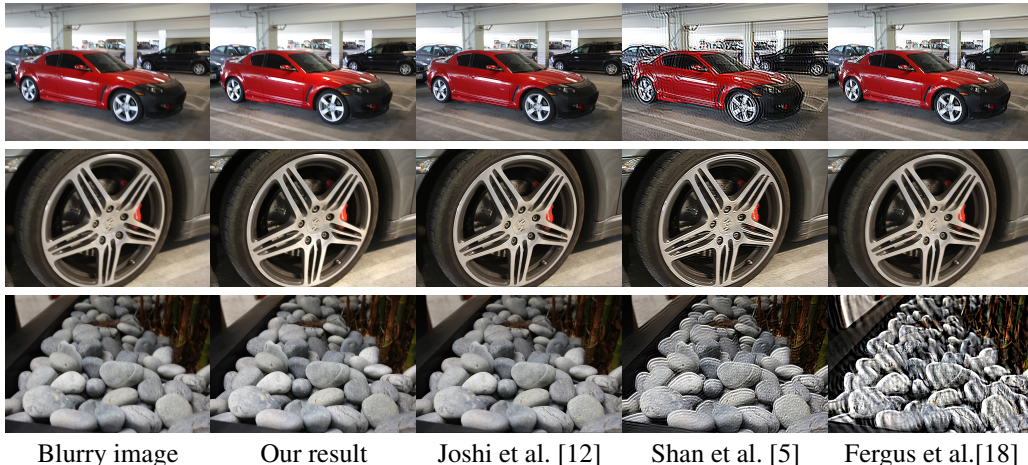

| Blurry image | Our result | Joshi et al. [12] | Shan et al. [5] | Fergus et al.[18] |

Figure 4: Our blind method achieves results comparable to Joshi et al. [12] who additionally require motion sensor information which we do not use. All images apart from our own algorithm's results are taken from [12]. This figure is best viewed on screen rather than in print.

**Comparison with Joshi et al.'s recent results**. Fig. 4 compares the results from [12] with our method on their example images. Even though our method does not exploit the motion sensor data utilized by Joshi et al. we obtain comparable results.

**Run-time.** The running times of our method is about 30 minutes for the images in Fig. 3 and about 80 minutes on the larger images of [12] ($1123 \times 749$ pixels in size). How does this compare with Cho and Lee's method for fast deblurring, which works in seconds? There are several reasons for the discrepancy: (i) Cho and Lee implemented their method using the GPU, while our implementation is in Matlab, logging lots of intermediate results for debugging and studying the code behaviour. (ii) A space-variant blur has more parameters, e.g. for 6 by 7 patches we need to estimate 42 times as many parameters as for a single kernel. Even though calculating the forward model is almost as fast as for the single kernel, convergence for that many parameters appeared to be slower. (iii) Cho and Lee are able to use direct deconvolution (division in Fourier space) for the image estimation step, while we have to solve an optimization problem, because we currently do not know how to perform direct deconvolution for the space-variant filters.

## 6 Discussion

Blind deconvolution of images degraded by *space-variant* blur is a much harder problem than simply assuming *space-invariant* blurs. Our experiments show that even state-of-the-art algorithms such as Cho and Lee's [2] are not able to recover image details for such blurs without unpleasant artifacts. We have proposed an algorithm that is able to tackle space-variant blurs with encouraging results.

Presently, the main limitation of our approach is that it can fail if the blurs are too large or if they vary too quickly across the image. We believe there are two main reasons for this: (i) on the one hand, if the blurs are large, the patches need to be large as well to obtain enough statistics for estimating the blur. On the other hand, if at the same time the PSF is varying too quickly, the patches need to be small enough. Our method only works if we can find a patch size and overlap setting that is a good trade-off for both requirements. (ii) The method of Cho and Lee [2], which is an important component of ours, does not work for all blurs. For instance, a PSF that looks like a thick horizontal line is challenging, because the resulting image feature might be misunderstood by the prediction step to be horizontal lines in the image. Improving the method of Cho and Lee [2] to deal with such blurs would be worthwhile.

Another limitation of our method are image areas with little structure. On such patches it is difficult to infer a reasonable blur kernel, and our method propagates the results from the neighboring patches to these cases. However, this propagation is heuristic and we hope to find a more rigorous approach to this problem in future work.

# References

[1] M. Hirsch, S. Sra, B. Schölkopf, and S. Harmeling. Efficient Filter Flow for Space-Variant Multiframe Blind Deconvolution. In *Proceedings of the IEEE Conference on Computer Vision and Pattern Recognition*, 2010.

[2] S. Cho and S. Lee. Fast Motion Deblurring. *ACM Transactions on Graphics (SIGGRAPH ASIA 2009)*, 28(5), 2009.

[3] D. Krishnan and R. Fergus. Fast image deconvolution using hyper-Laplacian priors. In *Advances in Neural Information Processing Systems (NIPS)*, 2009.

[4] R. Fergus, B. Singh, A. Hertzmann, S.T. Roweis, and W.T. Freeman. Removing camera shake from a single photograph. In *ACM SIGGRAPH*, page 794. ACM, 2006.

[5] Q. Shan, J. Jia, and A. Agarwala. High-quality motion deblurring from a single image. *ACM Transactions on Graphics (SIGGRAPH)*, 2008.

[6] D. Kundur and D. Hatzinakos. Blind image deconvolution. *IEEE Signal Processing Mag.*, 13(3):43–64, May 1996.

[7] A. Levin, Y. Weiss, F. Durand, and W.T. Freeman. Understanding and evaluating blind deconvolution algorithms. In *Proceedings of the IEEE Conference on Computer Vision and Pattern Recognition*, 2009.

[8] Y. W. Tai, P. Tan, L. Gao, and M. S. Brown. Richardson-Lucy deblurring for scenes under projective motion path. Technical report, KAIST, 2009.

[9] Qi Shan, Wei Xiong, and Jiaya Jia. Rotational motion deblurring of a rigid object from a single image. In *Proc. Int. Conf. on Computer Vision*, 2007.

[10] J. Bardsley, S. Jeffries, J. Nagy, and B. Plemmons. A computational method for the restoration of images with an unknown, spatially-varying blur. *Optics Express*, 14(5):1767–1782, 2006.

[11] J.G. Nagy and D.P. O'Leary. Restoring images degraded by spatially variant blur. *SIAM Journal on Scientific Computing*, 19(4):1063–1082, 1998.

[12] N. Joshi, S.B. Kang, C.L. Zitnick, and R. Szeliski. Image deblurring using inertial measurement sensors. In *ACM SIGGRAPH 2010 Papers*. ACM, 2010.

[13] A. Levin. Blind motion deblurring using image statistics. In *Advances in Neural Information Processing Systems (NIPS)*, 2006.

[14] S. Cho, Y. Matsushita, and S. Lee. Removing non-uniform motion blur from images. In *IEEE 11th International Conference on Computer Vision, 2007*, 2007.

[15] M. Šorel and F. Šroubek. Space-variant deblurring using one blurred and one underexposed image. In *Proceedings of the International Conference on Image Processing (ICIP)*, 2009.

[16] T.G. Stockham Jr. High-speed convolution and correlation. In *Proceedings of the Spring joint computer conference*, pages 229–233. ACM, 1966.

[17] N. Joshi, R. Szeliski, and D.J. Kriegman. Image/video deblurring using a hybrid camera. In *Proceedings of the IEEE Conference on Computer Vision and Pattern Recognition*, 2008.

[18] R. Fergus, B. Singh, A. Hertzmann, S.T. Roweis, and W.T. Freeman. Removing camera shake from a single image. *ACM Transactions on Graphics (SIGGRAPH)*, 2006.

